# Linear Hinge Loss and Average Margin

**Claudio Gentile**
DSI, Universita' di Milano,
Via Comelico 39,
20135 Milano. Italy
gentile@dsi.unimi.it

**Manfred K. Warmuth**[*]
Computer Science Department,
University of California,
95064 Santa Cruz, USA
manfred@cse.ucsc.edu

## Abstract

We describe a unifying method for proving relative loss bounds for on-line linear threshold classification algorithms, such as the Perceptron and the Winnow algorithms. For classification problems the discrete loss is used, i.e., the total number of prediction mistakes. We introduce a continuous loss function, called the "linear hinge loss", that can be employed to derive the updates of the algorithms. We first prove bounds w.r.t. the linear hinge loss and then convert them to the discrete loss. We introduce a notion of "average margin" of a set of examples . We show how relative loss bounds based on the linear hinge loss can be converted to relative loss bounds i.t.o. the discrete loss using the average margin.

## 1 Introduction

Consider the classical Perceptron algorithm. The hypothesis of this algorithm at trial $t$ is a linear threshold function determined by a weight vector $w_t \in \mathcal{R}^n$. For an instance $x_t \in \mathcal{R}^n$ the linear activation $\hat{a}_t = w_t \cdot x_t$ is passed through a threshold function $\sigma_r$ which is $-1$ on arguments less than the threshold $r$ and $+1$ otherwise. Thus the prediction of the algorithm is binary and $-1, +1$ denote the two classes. The Perceptron algorithm is aimed at learning a classification problem where the examples have the form $(x_t, y_t) \in \mathcal{R}^n \times \{-1, +1\}$.

After seeing $T$ examples $(x_t, y_t)_{1 \le t \le T}$, the algorithm predicts with $\hat{y}_{T+1} = \sigma_r(w_{T+1} \cdot x_{T+1})$ on the next instance $x_{T+1}$. If the algorithm's prediction $\hat{y}_{T+1}$ agrees with the label $y_{T+1}$ on the instance $x_{T+1}$, then its loss is zero. If the prediction and the label disagree, then the loss is one. We call this loss the discrete loss.

The convergence of the Perceptron algorithm is established in the Perceptron convergence theorem. There is a second by now classical algorithm for learning with linear threshold functions: the Winnow algorithm of Nick Littlestone [Lit88]. This algorithm also maintains a weight vector and predicts with the same linear threshold function defined by the current weight vector $w_t$. However, the update of the weight vector $w_t = (w_{t.1}, ..., w_{t.n})$

---

[*]Supported by NSF grant CCR-9700201.

performed by the two algorithms is radically different:

$$\text{Perceptron: } w_{t+1} := w_t - \eta\, \delta_t\, x_t$$
$$\text{Winnow: } \ln w_{t+1,i} := \ln w_{t,i} - \eta\, \delta_t\, x_{t,i}$$

The Perceptron algorithm performs a simple additive update. The parameter $\eta$ is a positive learning rate and $\delta_t$ equals $(\hat{y}_t - y_t)/2$, which lies in $\{-1, 0, +1\}$. When $\delta_t = 0$ the prediction of the algorithm is correct and no update occurs. Both the Perceptron algorithm and Winnow update *conservatively*, i.e., they update only when the prediction of the algorithm is wrong. If $\hat{y}_t = +1$ and $y_t = -1$ then the algorithm overshot and $\delta_t = +1$. This causes the Perceptron to subtract $\eta\, x_t$ from the current weight $w_t$. Similarly if $\hat{y}_t = -1$ and $y_t = +1$ then the algorithm undershot and $\delta_t = -1$. Now the Perceptron adds $\eta\, x_t$ to the current weight $w_t$. We will later interpret $\delta_t\, x_t$ as a gradient of a loss function. Winnow uses the same gradient but the update is done through the componentwise logarithm of the weight vector. One can also rewrite Winnow's update as

$$w_{t+1,i} := w_{t,i} \exp\left(-\eta\, \delta_t x_{t,i}\right), \quad i = 1, ..., n,$$

so that the gradient appears in the exponents of factors that multiply the old weights. The factors are now used to correct the weights in the right direction when the algorithm under or overshot.

The algorithms are good for different purposes and, generally speaking, incomparable (see [KWA97] for a discussion). In [KW97] a framework was introduced for deriving simple on-line learning updates. This framework has been applied to a variety of different learning algorithms and *differentiable* loss functions [HKW95, KW98]. The updates are always derived by approximately solving the following minimization problem

$$w_{t+1} := \operatorname{argmin}_w U(w), \text{ where } U(w) = d(w, w_t) + \eta\, \mathbf{loss}(y_t, \sigma_r(w \cdot x_t)). \quad (1)$$

Here **loss** denotes the chosen loss function. In our setting this would be the discrete loss. What is different now is that the prediction of the algorithm $\hat{y}_t = \sigma_r(w_t \cdot x_t)$ and the discrete loss are *discontinuous* in the weight vector $w_t$. We will return to this point later after discussing the other parts of the above minimization problem. The parameter $\eta$ is the learning rate mentioned above and, most importantly, $d(w, w_t)$ is a divergence measuring how far $w$ is from $w_t$. The divergence function has two purposes. It motivates the update and it becomes the potential function in the amortized analysis used to prove loss bounds for the corresponding algorithm.

The use of an amortized analysis in the context of learning essentially goes back to [Lit89] and the method for deriving updates based on the divergence was introduced in [KW97]. The divergence may be seen as a regularization term and may also serve as a barrier function in the optimization problem (1) for the purpose of keeping the weights in a particular region. The additive algorithms, such as gradient descent and the Perceptron algorithm, use $d(w, w_t) = ||w - w_t||^2/2$ as the divergence. This can be used as a potential function for the proof of the Perceptron convergence theorem. Multiplicative update algorithms such as Winnow and various exponentiated gradient algorithms use entropy-based divergences as potential functions [HKW95, KW98]. The function $U$ in (1) is minimized by differentiating w.r.t. $w$. This works very well when the loss function is convex and differentiable. For example for linear regression, when the loss function is the square loss $(w_t \cdot x_t - y_t)^2/2$, then minimizing $U(w)$ with the divergence $||w - w_t||^2/2$ gives the Widrow-Hoff update:

$$w_{t+1} := w_t - \eta(w_{t+1} \cdot x_t - y_t)x_t \approx w_t - \eta(w_t \cdot x_t - y_t)x_t.$$

Various exponentiated gradient algorithms [KW97] can be derived in the same way when entropic divergences are used instead. However, in our case we cannot differentiate the discrete loss since it is discontinuous.

We asked ourselves which loss function motivates the Perceptron and Winnow algorithms in this framework. We will see that the loss function that achieves this is continuous and

its gradient w.r.t. $w_t$ is $\delta_t x_t$, where $\delta_t \in \{-1, 0, +1\}$. We call this loss the (linear) *hinge loss* (HL) and we believe this is the key tool for understanding linear threshold algorithms such as the Perceptron and Winnow. However, in the process of changing the discrete loss to the HL we also changed our learning problem from a classification to a regression problem. There are now two versions of each algorithm, a classification version and a regression version. The classification version predicts with a binary label using its linearly thresholded prediction. The loss function is the discrete loss. The regression version, on the other hand, predicts on the next instance $x_t$ with its linear activation $\hat{a}_t = w_t \cdot x_t$. In the classification problem the labels $y_t$ of the examples are $-1$ and $+1$, while in the regression problem the labels $a_t$ are $-\infty$ and $+\infty$. We will see that both versions of each algorithm use the same rule to update the weight vector $w_t$.

Another strong hint that the HL is related to Perceptron and Winnow comes from the fact that this loss may be seen as a limiting case of the entropic loss used in logistic regression. In logistic regression the threshold function $\sigma_r$ is replaced by the smooth tanh function. There is a technical way of associating a "matching loss function" with a given increasing transfer function [HKW95]. The matching loss for the tanh transfer function is the entropic loss. We will show that by making this transfer function steeper and by taking the right viewpoint of the matching loss, the entropic loss converges to the HL. In the limiting case the slope of the transfer function is infinite, i.e., it becomes the threshold function $\sigma_r$.

The question is whether this introduction of the HL buys us anything. We believe so. We can prove a unifying meta-theorem for the whole class of *general additive algorithms* [GLS97, KW98], when defined w.r.t. the HL. The bounds for the regression versions of the Perceptron and Winnow are simple special cases. These loss bounds can then be converted to loss bounds for the corresponding classification problems w.r.t. the discrete loss. This conversion is carried out through working with the "average margin" of a set of examples relative to a linear threshold classifier. The conversion of the HL described in this paper can then be considered a principled way of deriving average margin-based mistake bounds. The average margin reveals the inner structure of mistake bound results that have been proven thus far for conservative on-line algorithms. Previously used definitions, such as the deviation [FS98] and the attribute error [Lit91], can easily be related to the average margin or reinterpreted in terms of the HL and the average margin.

## 2 Preliminaries and the linear hinge loss

We define two subsets of $\mathcal{R}^n$: the *weight domain* $\mathcal{W}$ and the *instance domain* $\mathcal{X}$. The weights $w$ maintained by the algorithms always lie in the weight domain and the instances $x$ of the examples always lie in the instance domain. We require $\mathcal{W}$ be convex.

A general additive algorithm and a divergence are defined in terms of a *link function* $\mathbf{f}$. Such a function is a vector valued function from the interior int $\mathcal{W}$ of the weight domain $\mathcal{W}$ onto $\mathcal{R}^n$, with the property that its Jacobian is strictly positive definite everywhere in int $\mathcal{W}$. A link function $\mathbf{f}$ has a unique inverse $\mathbf{f}^{-1} : \mathcal{R}^n \to$ int $\mathcal{W}$. We assume that $\mathbf{f}$ is the gradient of a (potential) function $P_{\mathbf{f}}$ from int $\mathcal{W}$ to $\mathcal{R}$, i.e., $\mathbf{f}(w) = \nabla P_{\mathbf{f}}(w)$ for $w \in$ int $\mathcal{W}$. It is easy to extend the domain of $P_{\mathbf{f}}$ such that it includes the boundary of $\mathcal{W}$.

For any link function $\mathbf{f}$, a *(Bregman) divergence* function $d_{\mathbf{f}} : \mathcal{W} \times$ int $\mathcal{W} \to [0, \infty)$ is defined as [Bre67]:

$$d_{\mathbf{f}}(u, w) = P_{\mathbf{f}}(u) - P_{\mathbf{f}}(w) - (u - w) \cdot \mathbf{f}(w). \tag{2}$$

Thus $d_{\mathbf{f}}(u, w)$ is the difference between $P_{\mathbf{f}}(u)$ and its first order Taylor expansion around $w$. Since $\mathbf{f}$ has a strictly positive definite Jacobian everywhere in int $\mathcal{W}$, the potential $P_{\mathbf{f}}$ is strictly convex over $\mathcal{W}$. Thus $d_{\mathbf{f}}(u, w) \geq 0$ with equality holding iff $u = w$.

The Perceptron algorithm is motivated by the identity link $\mathbf{f}(w) = w$, with weight domain $\mathcal{W} = \mathcal{R}^n$. The corresponding divergence is $d_{\mathbf{f}}(u, w) = ||u - w||^2 / 2$. For Winnow the

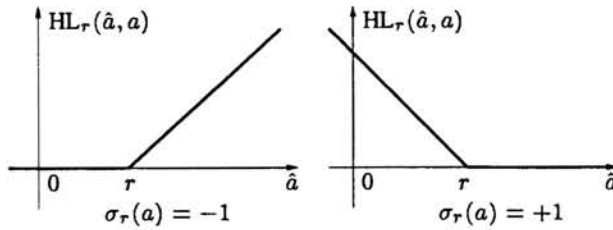

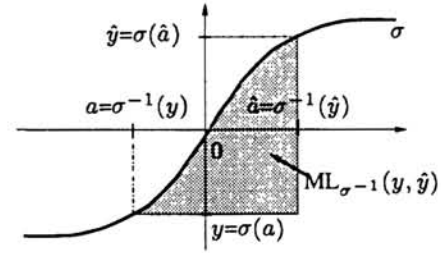

Figure 1: $\mathrm{HL}(\hat{a}, a)$ as a function of $\hat{a}$ for the two cases $\sigma_r(a) = -1, +1$.

Figure 2: The matching loss $\mathrm{ML}_{\sigma^{-1}}(y, \hat{y})$.

weight domain is $\mathcal{W} = [0, \infty)^n$. The link function is the componentwise logarithm. The divergence related to this link function is the un-normalized relative entropy $d_f(u, w) = \sum_{i=1}^n u_i \ln \frac{u_i}{w_i} + w_i - u_i$. Note that now $u \in \mathcal{W}$, but $w$ must lie in int $\mathcal{W}$.

The following key property immediately follows from the definition of the divergence $d_f$.

**Lemma 1** *[KW98] For any $u \in \mathcal{W}$ and $w_1, w_2 \in int \; \mathcal{W}$:*

$$d_f(u, w_2) - d_f(u, w_1) - d_f(w_1, w_2) = (u - w_1) \cdot (\mathbf{f}(w_1) - \mathbf{f}(w_2)).$$

In this paper we focus on a single neuron using a hard threshold as the transfer function (see beginning of the introduction). We will view such a neuron in two ways. In the standard view the neuron is used for binary classification. It outputs $\hat{y} = \sigma_r(\hat{a})$ trying to predict the desired label $y$ using a threshold $r$. In the new view the neuron is a regressor. It outputs the linear activation $\hat{a} \in \mathcal{R}$ and is trying to predict $a \in \mathcal{R}$.

For classification we use the discrete loss $\mathrm{DL}(y, \hat{y}) = \frac{1}{2}|\hat{y} - y| \in \{0, 1\}$. For regression we use the linear hinge loss (HL) parameterized by a threshold $r$:

$$\text{For any } a, \hat{a} \in \mathcal{R}: \quad \mathrm{HL}_r(\hat{a}, a) := \tfrac{1}{2}(\sigma_r(\hat{a}) - \sigma_r(a))(\hat{a} - r) = \mathrm{DL}(y, \hat{y})|\hat{a} - r|.$$

Note that the arguments in the two losses DL and $\mathrm{HL}_r$ are switched. This is intentional and will be discussed later on.

It can be easily shown that $\mathrm{HL}_r(w \cdot x, a)$ is convex in $w$ and that the gradient of this loss w.r.t. $w$ is $\nabla_w \mathrm{HL}_r(w \cdot x, a) = \frac{1}{2}(\sigma_r(\hat{a}) - \sigma_r(a)) x$. Note that $\delta = (\sigma_r(\hat{a}) - \sigma_r(a))/2$ can only take the three values $0$, $-1$, and $+1$ mentioned in the introduction. Strictly speaking, this gradient is not defined when $w \cdot x$ equals the threshold $r$. But we will show in the subsequent sections that even in that case $\delta\, x$ has the properties we need. Figure 1 provides a graphical representation of $\mathrm{HL}_r$. The threshold function $\sigma_r$ "transfers" the linear activation $\hat{a} = w \cdot x$ to a prediction $\hat{y}$ which is a hard classification in $\{-1, +1\}$. (For the remaining discussion of this section we can assume with no loss of generality that the threshold $r$ is 0.) Smooth transfer functions such as the tanh are commonly used in neural networks, e.g., $\hat{y} = \tanh(\hat{a})$, and relative loss bounds have been proven when the comparison class consists of single neurons with any increasing differentiable transfer function $\sigma$ [HKW95, KW98]. However, for this to work a loss function that "matches" the transfer function has to be used. This loss is defined[1] as follows [HKW95] (see Figure 2):

$$\mathrm{ML}_{\sigma^{-1}}(y, \hat{y}) := \int_{\sigma^{-1}(y)}^{\sigma^{-1}(\hat{y})} \sigma(z) - y \; dz = d_{\sigma^{-1}}(y, \hat{y}).$$

The matching loss for $\sigma(z) = z$ is the square loss (linear regression) and the matching loss for $\sigma(z) = \tanh(z)$ is the entropic loss (logistic regression), which is defined as:

$\text{ML}_{\sigma^{-1}}(y, \hat{y}) = \frac{1}{2}(1 - y)\ln\frac{1-y}{1-\hat{y}} + \frac{1}{2}(1 + y)\ln\frac{1+y}{1+\hat{y}}$. The entropic loss is finite when $y \in [-1, +1]$ and $\hat{y} = \tanh(\hat{a}) \in (-1, +1)$. These are the ranges for $y$ and $\hat{y}$ needed for logistic regression. We now want to use this type of loss for classification with linear threshold functions, i.e., when $y, \hat{y} \in \{-1, +1\}$ and the slope $s$ of the $\mathtt{tanh}$ function is increased until in the limit it becomes the hard threshold $\sigma_0$. Obviously, $\sigma^{-1}(-1) = -\infty$ and $\sigma^{-1}(+1) = +\infty$ for any slope $s$. Thus the matching loss is infinite for all slopes. Also, the known relative loss bounds based on the above notion of matching loss grow with the slope of the transfer function. Thus it seems to be impossible to use the matching loss when the transfer function is the hard threshold $\sigma_0$. However, we can still make sense of the matching loss by viewing the neuron as a regressor. The matching loss is now rewritten as another Bregman divergence:

$$\widetilde{\text{ML}}_\sigma(\hat{a}, a) = \int_a^{\hat{a}} \sigma(z) - \sigma(a) \; dz = P_\sigma(\hat{a}) - P_\sigma(a) - (\hat{a} - a)\sigma(a) = d_\sigma(\hat{a}, a), \quad (3)$$

where $P_\sigma$ is any function such that $P_\sigma'(a) = \sigma(a)$. We now increase the slope of the transfer function $\mathtt{tanh}$ while keeping $\hat{a}$ and $a$ fixed. In the limiting case (hard threshold $\sigma_0$) the above loss becomes twice the linear hinge loss with threshold zero, i.e., $\widetilde{\text{ML}}_{\sigma_0}(\hat{a}, a) = 2\,\text{HL}_0(\hat{a}, a) = (\sigma_0(\hat{a}) - \sigma_0(a))(\hat{a} - 0)$. Finally, observe that the two views of the neuron are related to a duality property [AW98] of Bregman divergences:

$$d_{\sigma^{-1}}(y, \hat{y}) = \text{ML}_{\sigma^{-1}}(y, \hat{y}) = \widetilde{\text{ML}}_\sigma(\hat{a}, a) = d_\sigma(\hat{a}, a). \quad (4)$$

## 3   The algorithms

In this paper we always associate two general additive algorithms with a given link function: a classification algorithm and a regression algorithm. Such algorithms, given in the next table, correspond to the two views of a linear threshold neuron discussed in the last section. For brevity, we will call the two algorithms "the classification algorithm" and "the regression algorithm", respectively.

| **Gen. add. classification algorithm:** | **Gen. add. regression algorithm:** |
|---|---|
| For $t = 1, 2, \ldots$ | For $t = 1, 2, \ldots$ |
| Instance: $x_t \in \mathcal{R}^n$ | Instance: $x_t \in \mathcal{R}^n$ |
| Prediction: $\hat{y}_t = \sigma_r(w_t \cdot x_t)$ | Prediction: $\hat{a}_t = w_t \cdot x_t$ |
| Label: $y_t \in \{-1, +1\}$ | Label:[2] $a_t = y_t\infty$ |
| Update: | Update: |
| $w_{t+1} = \mathbf{f}^{-1}\left(\mathbf{f}(w_t) - \frac{\eta}{2}(\hat{y}_t - y_t)x_t\right)$ | $w_{t+1} = \mathbf{f}^{-1}\left(\mathbf{f}(w_t) - \frac{\eta}{2}(\sigma_r(\hat{a}_t) - \sigma_r(a_t))x_t\right)$ |
| Discrete loss: | Linear hinge loss: |
| $\text{DL}(y_t, \hat{y}_t) = \frac{1}{2}\|\hat{y}_t - y_t\|$ | $\text{HL}_r(\hat{a}_t, a_t) = \frac{1}{2}(\sigma_r(\hat{a}_t) - \sigma_r(a_t))(\hat{a}_t - r)$ |

The classification algorithm receives a label $y_t \in \{-1, +1\}$, while the regression algorithm receives the infinite label $a_t$ with the sign of $y_t$. This assures that $y_t = \sigma_r(a_t)$. The classification algorithm predicts with $\hat{y}_t = \sigma_r(\hat{a}_t)$, and the regression algorithm with its linear activation $\hat{a}_t$. The loss for the classification algorithm is the discrete loss $\text{DL}(y_t, \hat{y}_t)$, while for the regression algorithm we use $\text{HL}_r(\hat{a}_t, a_t)$. The updates of the two algorithms are equivalent. The update of the regression algorithm is motivated by the minimization problem:

$$w_{t+1} := \text{argmin}_w U(w) \text{ where } U(w) = d_\mathbf{f}(w, w_t) + \eta\,\text{HL}_r(w \cdot x_t, a_t).$$

By setting the gradient of $U(w)$ w.r.t. $w$ to zero we get the following equilibrium equation that holds at the minimum of $U(w)$: $w_{t+1} = \mathbf{f}^{-1}\left(\mathbf{f}(w_t) - \frac{\eta}{2}(\sigma_r(w_{t+1} \cdot x_t) - \sigma_r(a_t))x_t\right)$. We approximately solve this equation by replacing $w_{t+1} \cdot x_t$ by $\hat{a}_t = w_t \cdot x_t$, i.e., $w_{t+1} = \mathbf{f}^{-1}\left(\mathbf{f}(w_t) - \frac{\eta}{2}(\sigma_r(\hat{a}_t) - \sigma_r(a_t))x_t\right)$.

Both versions of the Perceptron and Winnow are obtained by using the link functions $\mathbf{f}(w) = w$ and $\mathbf{f}(w) = (\ln(w_1), ..., \ln(w_n))$, respectively.

## 4  Relative loss bounds

The following lemma relates the hinge loss of the regression algorithm to the hinge loss of an arbitrary linear predictor $u$.

**Lemma 2** *For all $u \in \mathcal{W}$, $w_t \in int\ \mathcal{W}$, $x_t \in \mathcal{X}$, $a_t, r \in \mathcal{R}$ and $\eta > 0$:*

$$\mathrm{HL}_r(\hat{a}_t, a_t) - \mathrm{HL}_r(u \cdot x_t, a_t) + \mathrm{HL}_r(u \cdot x_t, \hat{a}_t)$$
$$= \tfrac{1}{\eta}\left(d_\mathbf{f}(u, w_t) - d_\mathbf{f}(u, w_{t+1}) + d_\mathbf{f}(w_t, w_{t+1})\right) = \tfrac{1}{2}(\hat{y}_t - y_t)(\hat{a}_t - u \cdot x_t) \quad (5)$$

*Proof.* We have $d_\mathbf{f}(u, w_t) - d_\mathbf{f}(u, w_{t+1}) + d_\mathbf{f}(w_t, w_{t+1}) = (u - w_t) \cdot (f(w_{t+1}) - f(w_t)) = (w_t - u) \cdot \tfrac{\eta}{2}(\sigma_r(\hat{a}_t) - \sigma_r(a_t))\, x_t = \tfrac{\eta}{2}(\sigma_r(\hat{a}_t) - \sigma_r(a_t))(\hat{a}_t - u \cdot x_t) = \eta\left(\mathrm{HL}_r(\hat{a}_t, a_t) - \mathrm{HL}_r(u \cdot x_t, a_t) + \mathrm{HL}_r(u \cdot x_t, \hat{a}_t)\right)$. The first equality follows Lemma 1 and the second follows from the update rule of the regression algorithm. The last equality uses $\mathrm{HL}_r(\hat{a}_t, a_t)$ as a divergence $d_{\sigma_r}(\hat{a}_t, a_t)$ (see (4)) and again Lemma 1. $\square$

By summing the first equality of (5) over all trials $t$ we could relate the total $\mathrm{HL}_r$ of the regression algorithm to the total $\mathrm{HL}_r$ of the regressor $u$. However, our goal is to obtain bounds on the number of mistakes on the classification algorithm. It is therefore natural to *interpret* $u$ too as a linear threshold classifier, with the same threshold $r$ used by the classification algorithm. We use the second equality of (5) and sum up over all $T$ trials:

$$\sum_{t=1}^T \tfrac{1}{2}(\hat{y}_t - y_t)\,(\hat{a} - u \cdot x_t) = \tfrac{1}{\eta}\left(d_\mathbf{f}(u, w_1) - d_\mathbf{f}(u, w_{T+1}) + \sum_{t=1}^T d_\mathbf{f}(w_t, w_{t+1})\right).$$

Note that the sums in the above equality are unaffected by trials in which no mistake occurs. In such trials, $\hat{y}_t = y_t$ and $w_{t+1} = w_t$. Thus the above is equivalent to the following, where $\mathcal{M}$ is the set of trials in which a mistake occurs:

$$\sum_{t \in \mathcal{M}} \tfrac{1}{2}(\hat{y}_t - y_t)\,(\hat{a}_t - u \cdot x_t) = \tfrac{1}{\eta}\left(d_\mathbf{f}(u, w_1) - d_\mathbf{f}(u, w_{T+1}) + \sum_{t \in \mathcal{M}} d_\mathbf{f}(w_t, w_{t+1})\right).$$

Since $\tfrac{1}{2}(\hat{y}_t - y_t) = -y_t$ when $t \in \mathcal{M}$ and $d_\mathbf{f}(u, w_{T+1}) \geq 0$ we get the following theorem:

**Theorem 3** *Let $\mathcal{M} \subseteq \{1, \ldots, T\}$ be the set of trials in which the classification algorithm makes a mistake. Then for every $u \in \mathcal{W}$ we have*

$$\sum_{t \in \mathcal{M}} y_t\,(u \cdot x_t - \hat{a}_t) \leq \tfrac{1}{\eta}\left(d_\mathbf{f}(u, w_1) + \sum_{t \in \mathcal{M}} d_\mathbf{f}(w_t, w_{t+1})\right). \square$$

Throughout the rest of this section the classification algorithm is compared to the performance of a linear threshold classifier $u$ with threshold $r = 0$. We now apply Theorem 3 to the Perceptron algorithm with $w_1 = 0$, giving a bound i.t.o. the *average margin* of a linear threshold classifier $u$ with threshold 0 on a trial sequence $\mathcal{M}$:

$$\hat{\gamma}_{u,\mathcal{M}} := \tfrac{1}{|\mathcal{M}|} \sum_{t \in \mathcal{M}} y_t u \cdot x_t.$$

Since $y_t \hat{a}_t \leq 0$ for $t \in \mathcal{M}$, the l.h.s. of the inequality of Theorem 3 is at least $|\mathcal{M}|\hat{\gamma}_{u,\mathcal{M}}$. By the update rule, $\sum_{t \in \mathcal{M}} d_\mathbf{f}(w_t, w_{t+1}) = \sum_{t \in \mathcal{M}} \tfrac{\eta^2}{2}\|x_t\|_2^2 \leq \tfrac{\eta^2}{2}|\mathcal{M}|X_2^2$, where $\|x\|_2 \leq X_2$ for $t \in \mathcal{M}$. Since in Theorem 3 $u$ is an arbitrary vector, we replace $u$ by $\lambda u$ therein, and set $\lambda = \frac{X_2^2 \eta}{\hat{\gamma}_{u,\mathcal{M}}}$. When we solve the resulting inequality for $|\mathcal{M}|$ the dependence on $\eta$ cancels out. This gives us the following bound on the number of mistakes:

$$|\mathcal{M}| \leq \left(\frac{\|u\|_2 X}{\hat{\gamma}_{u,\mathcal{M}}}\right)^2.$$

Note that in the usual mistake bound for the Perceptron algorithm the average $\hat{\gamma}_{u,\mathcal{M}}$ is replaced by $\min_{t \in \mathcal{M}} y_t u \cdot x_t$.[3] Also, observe that the predictions of the Perceptron algorithm with $r = 0$ and $w_1 = 0$ are not affected by $\eta$. Hence the previous bound holds for any $\eta > 0$.

Next, we apply Theorem 3 to a normalized version of Winnow. This version of Winnow keeps weights in the probability simplex and is obtained by a slight modification of Winnow's link function. We assume $r = 0$ and choose $\mathcal{X} = \{x \in \mathcal{R}^n : ||x||_\infty \leq X_\infty\}$. Unlike the Perceptron algorithm, a Winnow-like algorithm heavily depends on the learning rate, so a careful tuning is needed. One can show (details omitted due to space limitations) that if $\eta$ is such that $\eta\,\hat{\gamma}_{u,\mathcal{M}} + \eta\,X_\infty - \ln\left(\frac{e^{2\eta X_\infty}+1}{2}\right) > 0$ then this normalized version of Winnow achieves the bound

$$|\mathcal{M}| \leq \frac{d_f(u, w_1)}{\eta\,\hat{\gamma}_{u,\mathcal{M}} + \eta\,X_\infty - \ln\left(\frac{e^{2\eta X_\infty}+1}{2}\right)},$$

where $d_f(u, w_1)$ is the relative entropy between the two probability vectors $u$ and $w_1$.

**Conclusions:** In the full paper we study the case when there is no consistent threshold $u$ more carefully and give more involved bounds for the Winnow and normalized Winnow algorithms as well as for the $p$-norm Perceptron algorithm [GLS97].

## Footnotes

[1] In [HKW95] the notation $L_\sigma(y, \hat{y})$ is used for the matching loss $\mathrm{ML}_{\sigma^{-1}}(y, \hat{y})$. We use here the subscript $\sigma^{-1}$ instead of $\sigma$ to stress a connection between the matching loss and the divergence that is discussed at the end of this section.

[2]This is a short-hand meaning $a_t = +\infty$ if $y_t = +1$ and $a_t = -\infty$ if $y_t = -1$.

[3]The average margin $\hat{\gamma}_{u,\mathcal{M}}$ may be positive even though $u$ is not consistent.

# References

[AW98]      K. Azoury and M. K. Warmuth", "Relative loss bounds and the exponential family of distributions", "1998", Unpublished manuscript.

[Bre67]     L.M. Bregman. The relaxation method of finding the common point of convex sets and its application to the solution of problems in convex programming. *USSR Computational Mathematics and Physics*, 7:200–217, 1967.

[FS98]      Y. Freund and R. Schapire. Large margin classification using the perceptron algorithm. In *11th COLT*, pp. 209–217, ACM, 1998.

[GLS97]     A. J. Grove, N. Littlestone, and D. Schuurmans. General convergence results for linear discriminant updates. In *10th COLT*, pp. 171–183. ACM, 1997.

[HKW95]  D. P. Helmbold, J. Kivinen, and M. K. Warmuth. Worst-case loss bounds for sigmoided linear neurons. In *NIPS 1995*, pp. 309–315. MIT Press, 1995.

[KW97]      J. Kivinen and M. K. Warmuth. Additive versus exponentiated gradient updates for linear prediction. *Inform. and Comput.*, 132(1):1–64, 1997.

[KW98]      J. Kivinen and M. K. Warmuth. Relative loss bounds for multidimensional regression problems. In *NIPS 10*, pp. 287–293. MIT Press, 1998.

[KWA97]  J. Kivinen, M. K. Warmuth, and P. Auer. The perceptron algorithm vs. winnow: linear vs. logarithmic mistake bounds when few input variables are relevant. *Artificial Intelligence*, 97:325–343, 1997.

[Lit88]      N. Littlestone. Learning when irrelevant attributes abound: A new linear-threshold algorithm. *Machine Learning*, 2:285–318, 1988.

[Lit89]      N. Littlestone. *Mistake Bounds and Logarithmic Linear-threshold Learning Algorithms*. PhD thesis, University of California Santa Cruz, 1989.

[Lit91]      N. Littlestone. Redundant noisy attributes, attribute errors, and linear threshold learning using Winnow. In *4th COLT*, pp. 147–156, Morgan Kaufmann, 1991.

